# Human and Machine 'Quick Modeling'

**Jakob Bernasconi**
Asea Brown Boveri Ltd
Corporate Research
CH–5405 Baden,
SWITZERLAND

**Karl Gustafson**
University of Colorado
Department of Mathematics and
Optoelectronic Computing Center
Boulder, CO 80309

## ABSTRACT

We present here an interesting experiment in 'quick modeling' by humans, performed independently on small samples, in several languages and two continents, over the last three years. Comparisons to decision tree procedures and neural net processing are given. From these, we conjecture that human reasoning is better represented by the latter, but substantially different from both. Implications for the 'strong convergence hypothesis' between neural networks and machine learning are discussed, now expanded to include human reasoning comparisons.

## 1 INTRODUCTION

Until recently the fields of symbolic and connectionist learning evolved separately. Suddenly in the last two years a significant number of papers comparing the two methodologies have appeared. A beginning synthesis of these two fields was forged at the NIPS '90 Workshop #5 last year (Pratt and Norton, 1990), where one may find a good bibliography of the recent work of Atlas, Dietterich, Omohundro, Sanger, Shavlik, Tsoi, Utgoff and others.

It was at that NIPS '90 Workshop that we learned of these studies, most of which concentrate on performance comparisons of decision tree algorithms (such as ID3, CART) and neural net algorithms (such as Perceptrons, Backpropagation). Independently three years ago we had looked at Quinlan's ID3 scheme (Quinlan, 1984) and intuitively and rather instantly not agreeing with the generalization he obtains by ID3 from a sample of 8 items generalized to 12 items, we subjected this example to a variety of human experiments. We report our findings, as compared to the performance of ID3 and also to various neural net computations.

Because our focus on humans was substantially different from most of the other mentioned studies, we also briefly discuss some important related issues for further investigation. More details are given elsewhere (Bernasconi and Gustafson, to appear).

## 2    THE EXPERIMENT

To illustrate his ID3 induction algorithm, Quinlan (1984) considers a set $C$ consisting of 8 objects, with attributes height, hair, and eyes. The objects are described in terms of their attribute values and classified into two classes, "+" and "−", respectively (see Table 1). The problem is to find a rule which correctly classifies all objects in $C$, and which is in some sense minimal.

Table 1: The set $C$ of objects in Quinlan's classification example.

| Object | Height | Hair | Eyes | Class |
|--------|--------|------|------|-------|
| 1 | (s) short | (b) blond | (bl) blue | + |
| 2 | (t) tall | (b) blond | (br) brown | − |
| 3 | (t) tall | (r) red | (bl) blue | + |
| 4 | (s) short | (d) dark | (bl) blue | − |
| 5 | (t) tall | (d) dark | (bl) blue | − |
| 6 | (t) tall | (b) blond | (bl) blue | + |
| 7 | (t) tall | (d) dark | (br) brown | − |
| 8 | (s) short | (b) blond | (br) brown | − |

The ID3 algorithm uses an information-theoretic approach to construct a "minimal" classification rule, in the form of a decision tree, which correctly classifies all objects in the learning set $C$. In Figure 1, we show two possible decision trees which correctly classify all 8 objects of the set $C$. Decision tree 1 is the one selected by the ID3 algorithm. As can be seen, "Hair" as root of the tree classifies four of the eight objects immediately. Decision tree 2 requires the same number of tests and has the same number of branches, but "Eyes" as root classifies only three objects at the first level of the tree.

Consider now how the decision trees of Figure 1 classify the remaining four possible objects in the set complement $C'$. Table 2 shows that the two decision trees lead to a different classification of the four objects of sample $C'$. We observe that the ID3-preferred decision tree 1 places a large importance on the "red" attribute (which occurs only in one object of sample $C$), while decision tree 2 puts much less emphasis on this particular attribute.

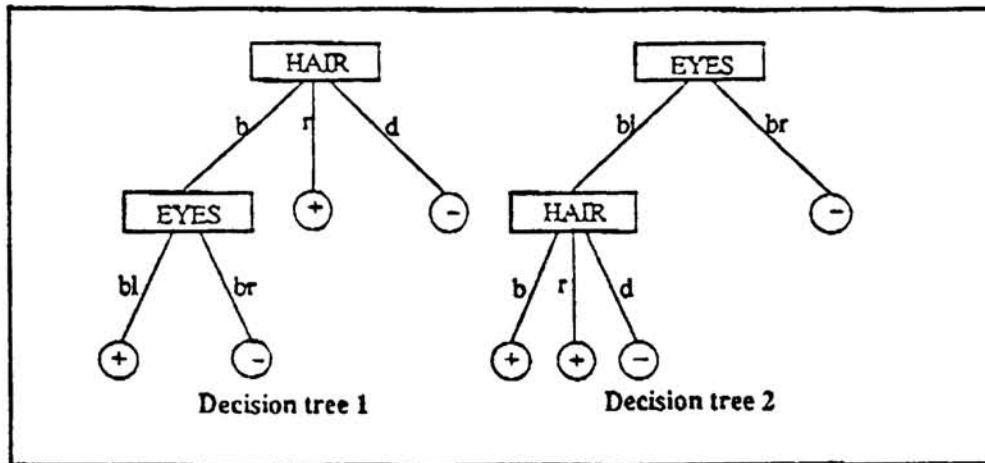

Figure 1: Two possible decision trees for the classification of sample C (Table 1)

Table 2: The set $C'$ of the remaining four objects, and their classification by the decision trees of Figure 1.

| Object | Attribute Values | | | Classification | |
|---|---|---|---|---|---|
| | | | | Tree 1 | Tree 2 |
| 9 | s | d | br | – | – |
| 10 | s | r | bl | + | + |
| 11 | s | r | br | + | – |
| 12 | t | r | br | + | – |

## 3    GENERALIZATIONS BY HUMANS AND NEURAL NETS

Curious about these differences in the generalization behavior, we have asked some humans (colleagues, graduate students, undergraduate students, some nonscientists also) to "look" at the original sample $C$ of 8 items, presented to them without warning, and to "use" this information to classify the remaining 4 objects. Over some time, we have accumulated a "human sample" of total size 73 from 3 continents representing 14 languages. The results of this human generalization experiment are summarized in Table 3. We observe that about 2/3 of the test persons generalized in the same manner as decision tree 2, and that less than 10 percent arrived at the generalization corresponding to the ID3-preferred decision tree 1.

Table 3: Classification of objects 9 through 12 by Humans and by a Neural Net. Based on a total sample of 73 humans. Each of the 4 contributing subsamples from different languages and locations gave consistent percentages.

| Object | Attribute Values | | | Classification | | | | | |
| --- | --- | --- | --- | --- | --- | --- | --- | --- | --- |
| | | | | A | B | C | D | E | Other |
| 9 | s | d | br | − | − | − | − | − | ⋮ |
| 10 | s | r | bl | + | + | − | + | + | ⋮ |
| 11 | s | r | br | − | + | − | − | + | ⋮ |
| 12 | t | r | br | − | + | − | + | − | ⋮ |
| | **Humans:** | | | 65.8% | 8.2% | 4.1% | 9.6% | — | 12.3% |
| | **Neural Net:** | | | 71.4% | 12.1% | 9.4% | 4.2% | 2.9% | — |

We also subjected this generalization problem to a variety of neural net computations. In particular, we analyzed a simple perceptron architecture with seven input units representing a unary coding of the attribute values (i.e., a separate input unit for each attribute value). The eight objects of sample $C$ (Table 1) were used as training examples, and we employed the perceptron learning procedure (Rumelhart and McClelland, 1986) for a threshold output unit. In our initial experiment, the starting weights were chosen randomly in $(-1, 1)$ and the learning parameter $h$ (the magnitude of the weight changes) was varied between 0.1 and 1. After training, the net was asked to classify the unseen objects 9 to 12 of Table 2. Out of the 16 possible classifications of this four object test set, only 5 were realized by the neural net (labelled A through E in Table 3). The percentage values given in Table 3 refer to a total of 9000 runs (3000 each for $h = 0.1, 0.5$, and $1.0$, respectively). As can be seen, there is a remarkable correspondence between the solution profile of the neural net computations and that of the human experiment.

## 4 BACKWARD PREDICTION

There exist many different rules which all correctly classify the given set $C$ of 8 objects (Table 1), but which lead to a different generalization behavior, i.e., to a different classification of the remaining objects 9 to 12 (see Tables 2 and 3). From a formal point of view, all of the 16 possible classifications of objects 9 to 12 are equally probable, so that no a priori criterion seems to exist to prefer one generalization over the other. We have nevertheless attempted to quantify the obviously ill-defined notion of "meaningful generalization". To estimate the relative "quality" of different classification rules, we propose to analyze the "backward prediction ability" of the respective generalizations. This is evaluated as follows. An appropriate learning method (e.g., neural nets) is used to construct rules which explain a given classification of objects 9 to 12, and these rules are applied to classify the initial set $C$ of 8 objects. The 16 possible generalizations can then be rated according to their "backward prediction accuracy" with respect to the original classification of

the sample $C$. We have performed a number of such calculations and consistently found that the 5 generalizations chosen by the neural nets in the forward prediction mode (cf. Table 3) have by far the highest backward prediction accuracy (on the average between 5 and 6 correct classifications). Their negations ("+" exchanged with "−"), on the other hand, predict only about 2 to 3 of the 8 original classifications correctly, while the remaining 6 possible generalizations all have a backward prediction accuracy close to 50% (4 out of 8 correct). These results, representing averages over 1000 runs, are given in Table 4.

Table 4: Neural Net backward prediction accuracy for the different classifications of objects 9 to 12.

| 9 | 10 | 11 | 12 | Backward prediction accuracy (%) |
|---|---|---|---|---|
| − | + | − | − | 76.0 |
| − | + | + | − | 71.2 |
| − | + | + | + | 71.1 |
| − | + | − | + | 67.9 |
| − | − | − | − | 61.9 |
| − | − | + | − | 52.6 |
| − | − | − | + | 52.5 |
| + | + | − | − | 52.5 |
| + | + | + | − | 47.4 |
| + | + | − | + | 47.3 |
| − | − | + | + | 47.0 |
| + | + | + | + | 37.2 |
| + | − | + | − | 31.7 |
| + | − | − | − | 30.1 |
| + | − | − | + | 28.3 |
| + | − | + | + | 23.6 |

In addition to Neural Nets, we have also used the ID3 method to evaluate the backward predictive power of different generalizations. This method generates fewer rules than the Neural Nets (often only a single one), but the resulting tables of backward prediction accuracies all exhibit the same qualitative features. As examples, we show in Figure 2 the ID3 backward prediction trees for two different generalizations, the ID3-preferred generalization which classifies the objects 9 to 12 as $(-++)$, and the Human and Neural Net generalization $(-+--)$. Both trees have a backward prediction accuracy of 75% (provided that "blond hair" in tree (a) is classified randomly).

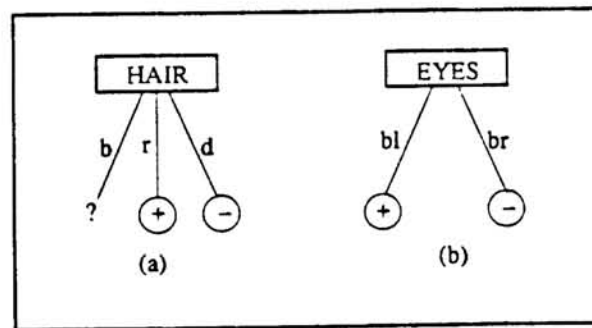

Figure 2: ID3 backward prediction trees, (a) for the ID3-preferred generalization
$(-+++)$, and (b) for the generalization preferred by Humans and Neural Nets,
$(-+--)$

The overall backward prediction accuracy is not the only quantity of interest in these
calculations. We can, for example, examine how well the original classification of an
individual object in the set $C$ is reproduced by predicting backwards from a given
generalization.

Some examples of such backward prediction profiles are shown in Figure 3. From
both the ID3 and the Neural Net calculations, it is evident that the backward
prediction behavior of the Human and Neural Net generalization is much more
informative than that of the ID3-solution, even though the two solutions have almost
the same average backward prediction accuracy.

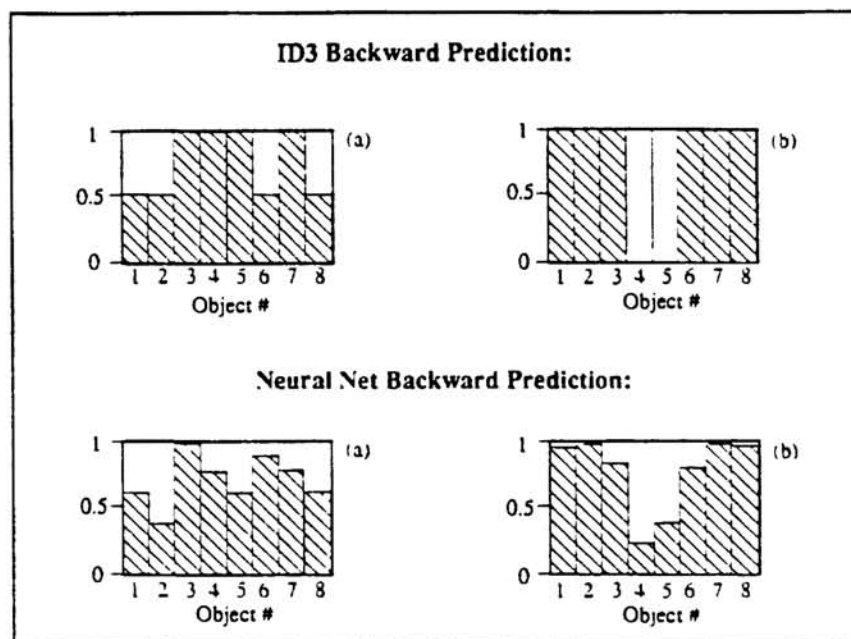

Figure 3: Individual backward prediction probabilities for the ID3-preferred gen-
eralization [graphs (a)], and for the Human and Neural Net generalization [graphs
(b)].

Finally, we have recently performed a Human backward prediction experiment. These results are given in Table 5. Details will be given elsewhere (Bernasconi and Gustafson, to appear). Note that the Backward Prediction results are commensurate with the Forward Prediction in both cases.

Table 5: Human backward predictions and accuracy from the two principal forward generalizations A (Neural Nets, Humans) and B (ID3).

| Object | Class | Backward from A | | Backward from B | |
|---|---|---|---|---|---|
| 1 | + | + | + | + | − |
| 2 | − | − | − | + | − |
| 3 | + | + | + | + | + |
| 4 | − | + | − | − | − |
| 5 | − | + | − | − | − |
| 6 | + | + | + | + | − |
| 7 | − | − | − | − | − |
| 8 | − | − | − | + | − |
| Humans: | | 59% | 12% | 33% | 17% |
| Accuracy: | | 75% | 100% | 75% | 75% |

# 5  DISCUSSION AND CONCLUSIONS

Our basic conclusion from this experiment is that the "Strong Convergence Hypothesis" that Machine Learning and Neural Network algorithms are "close" can be sharpened, with the two fields then better distinguished, by comparison to Human Modelling. From the experiment described here, we conjecture a "Stronger Convergence Hypothesis" that Humans and Neural Nets are "closer."

Further conclusions related to minimal network size (re Pavel, Gluck, Henkle, 1989), crossvalidation (see Weiss and Kulikowski, 1991), sharing over nodes (as in Dietterich, Hild, Bakiri, to appear, and Atlas et al., 1990), and rule extracting (Shavlik et al., to appear), will appear elsewhere (Bernasconi and Gustafson, to appear). Although we have other experiments on other test sets underway, it should be stressed that our investigations especially toward Human comparisons are only preliminary and should be viewed as a stimulus to further investigations.

**ACKNOWLEDGEMENT**

This work was partially supported by the NFP 23 program of the Swiss National Science Foundation and by the US–NSF grant CDR8622236.

## REFERENCES

L. Y. Pratt and S. W. Norton, "Neural Networks and Decision Tree Induction: Exploring the Relationship Between Two Research Areas," NIPS '90 Workshop #5 Summary (1990), 7 pp.

J. Ross Quinlan, "Learning Efficient Classification Procedures and Their Application to Chess End Games," in *Machine Learning: An Artificial Intelligence Approach*, edited by R. S. Michalski, J. G. Carbonell, and T. M. Mitchell, Springer-Verlag, Berlin (1984), 463–482.

D. E. Rumelhart and J. L. McClelland (Eds.), *Parallel Distributed Processing*, Vol. 1 MIT Press, Cambridge, MA (1986).

J. Bernasconi and K. Gustafson, "Inductive Inference and Neural Nets," to appear.

J. Bernasconi and K. Gustafson, "Generalization by Humans, Neural Nets, and ID3," IJCNN–91–Seattle.

Y. H. Pao, *Adaptive Pattern Recognition and Neural Networks*, Addison Wesley (1989), Chapter 4.

M. Pavel, M. A. Gluck and V. Henkle, "Constraints on Adaptive Networks for Modelling Human Generalization," in *Advances in Neural Information Processing Systems* 1, edited by D. Touretzky, Morgan Kaufmann, San Mateo, CA (1989), 2–10.

S. Weiss and C. Kulikowski, *Computer Systems that Learn*, Morgan Kaufmann (1991).

T. G. Dietterich, H. Hild, and G. Bakiri, "A Comparison of ID3 and Backpropagation for English Text-to-Speech Mapping," *Machine Learning*, to appear.

L. Atlas, R. Cole, J. Connor, M. El-Sharkawi, R. Marks, Y. Muthusamy, E. Barnard, "Performance Comparisons Between Backpropagation Networks and Classification Trees on Three Real-World Applications," in *Advances in Neural Information Processing Systems* 2, edited by D. Touretzky, Morgan Kaufmann (1990), 622–629.

J. Shavlik, R. Mooney, G. Towell, "Symbolic and Neural Learning Algorithms: An Experimental Comparison (revised)," *Machine Learning*, (1991, to appear).
